# Fast Rates for Regularized Objectives

**Karthik Sridharan, Nathan Srebro, Shai Shalev-Shwartz**
Toyota Technological Institute — Chicago

## Abstract

We study convergence properties of empirical minimization of a stochastic strongly convex objective, where the stochastic component is linear. We show that the value attained by the empirical minimizer converges to the optimal value with rate $1/n$. The result applies, in particular, to the SVM objective. Thus, we obtain a rate of $1/n$ on the convergence of the SVM objective (with fixed regularization parameter) to its infinite data limit. We demonstrate how this is essential for obtaining certain type of oracle inequalities for SVMs. The results extend also to approximate minimization as well as to strong convexity with respect to an arbitrary norm, and so also to objectives regularized using other $\ell_p$ norms.

## 1  Introduction

We consider the problem of (approximately) minimizing a stochastic objective

$$F(\mathbf{w}) = \mathbb{E}_\theta \left[ f(\mathbf{w}; \theta) \right] \tag{1}$$

where the optimization is with respect to $\mathbf{w} \in \mathbf{W}$, based on an i.i.d. sample $\theta_1, \ldots, \theta_n$. We focus on problems where $f(\mathbf{w}; \theta)$ has a generalized linear form:

$$f(\mathbf{w}; \theta) = \ell(\langle \mathbf{w}, \phi(\theta) \rangle, \theta) + r(\mathbf{w}) . \tag{2}$$

The relevant special case is regularized linear prediction, where $\theta = (\mathbf{x}, y)$, $\ell(\langle \mathbf{w}, \phi(\mathbf{x}) \rangle, y)$ is the loss of predicting $\langle \mathbf{w}, \phi(\mathbf{x}) \rangle$ when the true target is $y$, and $r(\mathbf{w})$ is a regularizer.

It is well known that when the domain $\mathbf{W}$ and the mapping $\phi(\cdot)$ are bounded, and the function $\ell(z; \theta)$ is Lipschitz continuous in $z$, the empirical averages

$$\hat{F}(\mathbf{w}) = \hat{\mathbb{E}} \left[ f(\mathbf{w}; \theta) \right] = \tfrac{1}{n} \sum_{i=1}^{n} f(\mathbf{w}; \theta_i) \tag{3}$$

converge uniformly to their expectations $F(\mathbf{w})$ with rate $\sqrt{1/n}$. This justifies using the empirical minimizer

$$\hat{\mathbf{w}} = \arg \min_{\mathbf{w} \in \mathbf{W}} \hat{F}(\mathbf{w}), \tag{4}$$

and we can then establish convergence of $F(\hat{\mathbf{w}})$ to the population optimum

$$F(\mathbf{w}^\star) = \min_{\mathbf{w} \in W} F(\mathbf{w}) \tag{5}$$

with a rate of $\sqrt{1/n}$.

Recently, Hazan *et al* [1] studied an online analogue to this problem, and established that if $f(\mathbf{w}; \theta)$ is *strongly convex* in $\mathbf{w}$, the average online regret diminishes with a much faster rate, namely $(\log n)/n$. The function $f(\mathbf{w}; \theta)$ becomes strongly convex when, for example, we have $r(\mathbf{w}) = \frac{\lambda}{2} \|\mathbf{w}\|^2$ as in SVMs and other regularized learning settings.

In this paper we present an analogous "fast rate" for empirical minimization of a strongly convex stochastic objective. In fact, we do not need to assume that we perform the empirical minimization

exactly: we provide uniform (over all $\mathbf{w} \in \mathbf{W}$) guarantees on the population sub-optimality $F(\mathbf{w}) - F(\mathbf{w}^\star)$ in terms of the empirical sub-optimality $\hat{F}(\mathbf{w}) - \hat{F}(\hat{\mathbf{w}})$ with a rate of $1/n$. This is a stronger type of result than what can be obtained with an online-to-batch conversion, as it applies to any possible solution $\mathbf{w}$, and not only to some specific algorithmically defined solution. For example, it can be used to analyze the performance of approximate minimizers obtained through approximate optimization techniques. Specifically, consider $f(\mathbf{w}; \theta)$ as in (2), where $\ell(z; \theta)$ is convex and $L$-Lipschitz in $z$, the norm of $\phi(\theta)$ is bounded by $B$, and $r$ is $\lambda$-strongly convex. We show that for any $a > 0$ and $\delta > 0$, with probability at least $1 - \delta$, for all $\mathbf{w}$ (of arbitrary magnitude):

$$F(\mathbf{w}) - F(\mathbf{w}^\star) \le (1+a)(\hat{F}(\mathbf{w}) - \hat{F}(\hat{\mathbf{w}})) + O\left((1 + 1/a)\frac{L^2 B^2(\log(1/\delta))}{\lambda n}\right). \qquad (6)$$

We emphasize that here and throughout the paper the big-$O$ notation hides only fixed numeric constants.

It might not be surprising that requiring strong convexity yields a rate of $1/n$. Indeed, the connection between strong convexity, variance bounds, and rates of $1/n$, is well known. However, it is interesting to note the generality of the result here, and the simplicity of the conditions. In particular, we do not require any "low noise" conditions, nor that the loss function is strongly convex (it need only be weakly convex).

In particular, (6) applies, under no additional conditions, to the SVM objective. We therefore obtain convergence with a rate of $1/n$ for the SVM objective. This $1/n$ rate on the SVM objective is *always* valid, and does *not* depend on any low-noise conditions or on specific properties of the kernel function. Such a "fast" rate might seem surprising at a first glance to the reader familiar with the $1/\sqrt{n}$ rate on the expected loss of the SVM optimum. There is no contradiction here—what we establish is that although the loss might converge at a rate of $1/\sqrt{n}$, the SVM objective (regularized loss) always converges at a rate of $1/n$.

In fact, in Section 3 we see how a rate of $1/n$ on the objective corresponds to a rate of $1/\sqrt{n}$ on the loss. Specifically, we perform an oracle analysis of the optimum of the SVM objective (rather than of empirical minimization subject to a norm constraint, as in other oracle analyses of regularized linear learning), based on the existence of some (unknown) low-norm, low-error predictor $\mathbf{w}$.

Strong convexity is a concept that depends on a choice of norm. We state our results in a general form, for any choice of norm $\|\cdot\|$. Strong convexity of $r(\mathbf{w})$ must hold with respect to the chosen norm $\|\cdot\|$, and the data $\phi(\theta)$ must be bounded with respect to the dual norm $\|\cdot\|_*$, i.e. we must have $\|\phi(\theta)\|_* \le B$. This allows us to apply our results also to more general forms of regularizers, including squared $\ell_p$ norm regularizers, $r(\mathbf{w}) = \frac{\lambda}{2}\|\mathbf{w}\|_p^2$, for $p < 1 \le 2$ (see Corollary 2). However, the reader may choose to read the paper always thinking of the norm $\|\mathbf{w}\|$, and so also its dual norm $\|\mathbf{w}\|_*$, as the standard $\ell_2$-norm.

## 2 Main Result

We consider a generalized linear function $f : \mathbf{W} \times \Theta \to \mathbb{R}$, that can be written as in (2), defined over a closed convex subset $\mathbf{W}$ of a Banach space equipped with norm $\|\cdot\|$.

**Lipschitz continuity and boundedness** We require that the mapping $\phi(\cdot)$ is bounded by $B$, i.e. $\|\phi(\theta)\|_* \le B$, and that the function $\ell(z; \theta)$ is $L$-Lipschitz in $z \in \mathbb{R}$ for every $\theta$.

**Strong Convexity** We require that $F(\mathbf{w})$ is $\lambda$-strongly convex w.r.t. the norm $\|\mathbf{w}\|$. That is, for all $\mathbf{w}_1, \mathbf{w}_2 \in \mathbf{W}$ and $\alpha \in [0, 1]$ we have:

$$F(\alpha\mathbf{w}_1 + (1-\alpha)\mathbf{w}_2) \le \alpha F(\mathbf{w}_1) + (1-\alpha)F(\mathbf{w}_2) - \frac{\lambda}{2}\alpha(1-\alpha)\|\mathbf{w}_1 - \mathbf{w}_2\|^2 .$$

Recalling that $\mathbf{w}^\star = \arg\min_{\mathbf{w}} F(\mathbf{w})$, this ensures (see for example [2, Lemma 13]):

$$F(\mathbf{w}) \ge F(\mathbf{w}^\star) + \frac{\lambda}{2}\|\mathbf{w} - \mathbf{w}^\star\|^2 \qquad (7)$$

We require only that the expectation $F(\mathbf{w}) = \mathbb{E}[f(\mathbf{w}; \theta)]$ is strongly convex. Of course, requiring that $f(\mathbf{w}; \theta)$ is $\lambda$-strongly convex for all $\theta$ (with respect to $\mathbf{w}$) is enough to ensure the condition.

In particular, for a generalized linear function of the form (2) it is enough to require that $\ell(z; y)$ is convex in $z$ and that $r(\mathbf{w})$ is $\lambda$-strongly convex (w.r.t. the norm $\|\mathbf{w}\|$).

We now provide a faster convergence rate using the above conditions.

**Theorem 1.** *Let* $\mathbf{W}$ *be a closed convex subset of a Banach space with norm* $\|\cdot\|$ *and dual norm* $\|\cdot\|_*$ *and consider* $f(\mathbf{w}; \theta) = \ell(\langle \mathbf{w}, \phi(\theta) \rangle; \theta) + r(\mathbf{w})$ *satisfying the Lipschitz continuity, boundedness, and strong convexity requirements with parameters* $B$, $L$, *and* $\lambda$. *Let* $\mathbf{w}^\star, \hat{\mathbf{w}}, F(\mathbf{w})$ *and* $\hat{F}(\mathbf{w})$ *be as defined in* (1)-(5). *Then, for any* $\delta > 0$ *and any* $a > 0$, *with probability at least* $1 - \delta$ *over a sample of size* $n$, *we have that for all* $\mathbf{w} \in \mathbf{W}$: *(where* $[x]_+ = \max(x, 0)$*)*

$$F(\mathbf{w}) - F(\mathbf{w}^\star) \leq (1 + a)[\hat{F}(\mathbf{w}) - \hat{F}(\mathbf{w}^\star)]_+ + \frac{8\left(1 + \frac{1}{a}\right)L^2 B^2(32 + \log(1/\delta))}{\lambda n}$$

$$\leq (1 + a)(\hat{F}(\mathbf{w}) - \hat{F}(\hat{\mathbf{w}})) + \frac{8\left(1 + \frac{1}{a}\right)L^2 B^2(32 + \log(1/\delta))}{\lambda n} .$$

It is particularly interesting to consider regularizers of the form $r(\mathbf{w}) = \frac{\lambda}{2} \|\mathbf{w}\|_p^2$, which are $(p-1)\lambda$-strongly convex w.r.t. the corresponding $\ell_p$-norm [2]. Applying Theorem 1 to this case yields the following bound:

**Corollary 2.** *Consider an* $\ell_p$ *norm and its dual* $\ell_q$, *with* $1 < p \leq 2$, $\frac{1}{q} + \frac{1}{p} = 1$, *and the objective* $f(\mathbf{w}; \theta) = \ell(\langle \mathbf{w}, \phi(\theta) \rangle; \theta) + \frac{\lambda}{2} \|\mathbf{w}\|_p^2$, *where* $\|\phi(\theta)\|_q \leq B$ *and* $\ell(z; y)$ *is convex and L-Lipschitz in* $z$. *The domain is the entire Banach space* $\mathbf{W} = \ell_p$. *Then, for any* $\delta > 0$ *and any* $a > 0$, *with probability at least* $1 - \delta$ *over a sample of size* $n$, *we have that for all* $\mathbf{w} \in \mathbf{W} = \ell_p$ *(of any magnitude):*

$$F(\mathbf{w}) - F(\mathbf{w}^\star) \leq (1 + a)(\hat{F}(\mathbf{w}) - \hat{F}(\hat{\mathbf{w}})) + O\left(\frac{(1 + \frac{1}{a})L^2 B^2 \log(1/\delta)}{(p - 1)\lambda n}\right) .$$

Corollary 2 allows us to analyze the rate of convergence of the regularized risk for $\ell_p$-regularized linear learning. That is, training by minimizing the empirical average of:

$$f(\mathbf{w}; \mathbf{x}, y) = \ell(\langle \mathbf{w}, \mathbf{x} \rangle, y) + \frac{\lambda}{2} \|\mathbf{w}\|_p^2 \tag{8}$$

where $\ell(z, y)$ is some convex loss function and $\|x\|_q \leq B$. For example, in SVMs we use the $\ell_2$ norm, and so bound $\|x\|_2 \leq B$, and the hinge loss $\ell(z, y) = [1 - yz]_+$, which is 1-Lipschitz. What we obtain is a bound on how quickly we can minimize the expectation $F(\mathbf{w}) = \mathbb{E}\left[\ell(\langle \mathbf{w}, \mathbf{x} \rangle, y)\right] + \frac{\lambda}{2} \|\mathbf{w}\|_p^2$, i.e. the regularized empirical loss, or in other words, how quickly do we converge to the infinite-data optimum of the objective.

We see, then, that the SVM objective converges to its optimum value at a fast rate of $1/n$, without any special assumptions. This still doesn't mean that the expected loss $\mathcal{L}(\hat{\mathbf{w}}) = \mathbb{E}\left[\ell(\langle \hat{\mathbf{w}}, \mathbf{x} \rangle, y)\right]$ converges at this rate. This behavior is empirically demonstrated on the left plot of Figure 1. For each data set size we plot the excess expected loss $\mathcal{L}(\hat{\mathbf{w}}) - \mathcal{L}(\mathbf{w}^\star)$ and the sub-optimality of the regularized expected loss $F(\hat{\mathbf{w}}) - F(\mathbf{w}^\star)$ (recall that $F(\hat{\mathbf{w}}) = \mathcal{L}(\hat{\mathbf{w}}) + \frac{\lambda}{2} \|\hat{\mathbf{w}}\|^2$). Although the regularized expected loss converges to its infinite data limit, i.e. to the population minimizer, with rate roughly $1/n$, the expected loss $\mathcal{L}(\hat{\mathbf{w}})$ converges at a slower rate of roughly $\sqrt{1/n}$.

Studying the convergence rate of the SVM objective allows us to better understand and appreciate analysis of computational optimization approaches for this objective, as well as obtain oracle inequalities on the generalization loss of $\hat{\mathbf{w}}$, as we do in the following Section.

Before moving on, we briefly provide an example of applying Theorem 1 with respect to the $\ell_1$-norm. The bound in Corollary 2 diverges when $p \to 1$ and the Corollary is not applicable for $\ell_1$ regularization. This is because $\|\mathbf{w}\|_1^2$ is *not* strongly convex w.r.t. the $\ell_1$-norm. An example of a regularizer that *is* strongly convex with respect to the $\ell_1$ norm is the (unnormalized) entropy regularizer [3]: $r(\mathbf{w}) = \sum_{i=1}^{d} |\mathbf{w}_i| \log(|\mathbf{w}_i|)$. This regularizer is $1/B_{\mathbf{w}}^2$-strongly convex w.r.t. $\|\mathbf{w}\|_1$, as long as $\|\mathbf{w}\|_1 \leq B_{\mathbf{w}}$ (see [2]), yielding:

**Corollary 3.** *Consider a function* $f(\mathbf{w}; \theta) = \ell(\langle \mathbf{w}, \phi(\theta) \rangle; \theta) + \sum_{i=1}^{d} |\mathbf{w}_i| \log(|\mathbf{w}_i|)$, *where* $\|\phi(\theta)\|_\infty \leq B$ *and* $\ell(z; y)$ *is convex and L-Lipschitz in* $z$. *Take the domain to be the* $\ell_1$ *ball*

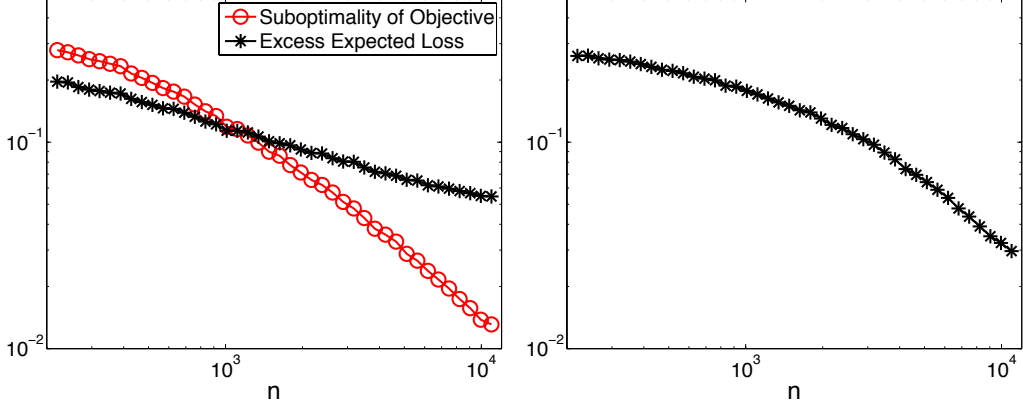

Figure 1: Left: Excess expected loss $\mathcal{L}(\hat{\mathbf{w}}) - \mathcal{L}(\mathbf{w}^\star)$ and sub-optimality of the regularized expected loss $F(\hat{\mathbf{w}}) - F(\mathbf{w}^\star)$ as a function of training set size, for a fixed $\lambda = 0.8$. Right: Excess expected loss $\mathcal{L}(\hat{\mathbf{w}}_\lambda) - \min_{\mathbf{w}} \mathcal{L}(\mathbf{w}_o)$, relative to the overall optimal $\mathbf{w}_o = \arg \min_{\mathbf{w}} \mathcal{L}(\mathbf{w})$, with $\lambda_n = \sqrt{300/n}$. Both plots are on a logarithmic scale and refer to a synthetic example with $\mathbf{x}$ uniform over $[-1.5, 1.5]^{300}$, and $y = \operatorname{sign} x_1$ when $|x_1| > 1$ but uniform otherwise.

$\mathbf{W} = \{\mathbf{w} \in \mathbb{R}^d : \|\mathbf{w}\|_1 \leq B_{\mathbf{w}}\}$. *Then, for any $\delta > 0$ and any $a > 0$, with probability at least $1 - \delta$ over a sample of size $n$, we have that for all $\mathbf{w} \in \mathbf{W}$:*

$$F(\mathbf{w}) - F(\mathbf{w}^\star) \leq (1 + a)(\hat{F}(\mathbf{w}) - \hat{F}(\hat{\mathbf{w}})) + O\left(\frac{(1 + \frac{1}{a})L^2 B^2 B_{\mathbf{w}}^2 \log(1/\delta)}{\lambda n}\right) \ .$$

## 3   Oracle Inequalities for SVMs

In this Section we apply the results from previous Section to obtain an oracle inequality on the expected loss $\mathcal{L}(\mathbf{w}) = \mathbb{E}\left[\ell(\langle \mathbf{w}, \mathbf{x} \rangle, y)\right]$ of an approximate minimizer of the SVM training objective $\hat{F}_\lambda(\mathbf{w}) = \hat{\mathbb{E}}\left[f_\lambda(\mathbf{w})\right]$ where

$$f_\lambda(\mathbf{w}; \mathbf{x}, y) = \ell(\langle \mathbf{w}, \mathbf{x} \rangle, y) + \frac{\lambda}{2} \|\mathbf{w}\|^2, \tag{9}$$

and $\ell(z, y)$ is the hinge-loss, or any other 1-Lipschitz loss function. As before we denote $B = \sup_{\mathbf{x}} \|\mathbf{x}\|$ (all norms in this Section are $\ell_2$ norms).

We assume, as an oracle assumption, that there exists a good predictor $\mathbf{w}_o$ with low norm $\|\mathbf{w}_o\|$ and which attains low expected loss $\mathcal{L}(\mathbf{w}_o)$. Consider an optimization algorithm for $\hat{F}_\lambda(\mathbf{w})$ that is guaranteed to find $\tilde{\mathbf{w}}$ such that $\hat{F}_\lambda(\tilde{\mathbf{w}}) \leq \min \hat{F}_\lambda(\mathbf{w}) + \epsilon_{\text{opt}}$. Using the results of Section 2, we can translate this approximate optimality of the empirical objective to an approximate optimality of the expected objective $F_\lambda(\mathbf{w}) = \mathbb{E}\left[f_\lambda(\mathbf{w})\right]$. Specifically, applying Corollary 2 with $a = 1$ we have that with probability at least $1 - \delta$:

$$F_\lambda(\tilde{\mathbf{w}}) - F_\lambda(\mathbf{w}^\star) \leq 2\epsilon_{\text{opt}} + O\left(\frac{B^2 \log(1/\delta)}{\lambda n}\right) \ . \tag{10}$$

Optimizing to within $\epsilon_{\text{opt}} = O(\frac{B^2}{\lambda n})$ is then enough to ensure

$$F_\lambda(\tilde{\mathbf{w}}) - F_\lambda(\mathbf{w}^\star) = O\left(\frac{B^2 \log(1/\delta)}{\lambda n}\right) \ . \tag{11}$$

In order to translate this to a bound on the expected loss $\mathcal{L}(\tilde{\mathbf{w}})$ we consider the following decomposition:

$$\mathcal{L}(\tilde{\mathbf{w}}) = \mathcal{L}(\mathbf{w}_o) + (F_\lambda(\tilde{\mathbf{w}}) - F_\lambda(\mathbf{w}^\star)) + (F_\lambda(\mathbf{w}^\star) - F_\lambda(\mathbf{w}_o)) + \frac{\lambda}{2} \|\mathbf{w}_o\|^2 - \frac{\lambda}{2} \|\tilde{\mathbf{w}}\|^2$$

$$\leq \mathcal{L}(\mathbf{w}_o) + O\left(\frac{B^2 \log(1/\delta)}{\lambda n}\right) + 0 + \frac{\lambda}{2} \|\mathbf{w}_o\|^2 \tag{12}$$

where we used the bound (11) to bound the second term, the optimality of $\mathbf{w}^\star$ to ensure the third term is non-positive, and we also dropped the last, non-positive, term.

This might seem like a rate of $1/n$ on the generalization error, but we need to choose $\lambda$ so as to balance the second and third terms. The optimal choice for $\lambda$ is

$$\lambda(n) = c \, \frac{B\sqrt{\log(1/\delta)}}{\|\mathbf{w}_o\| \, \sqrt{n}} \; , \tag{13}$$

for some constant $c$. We can now formally state our oracle inequality, which is obtained by substituting (13) into (12):

**Corollary 4.** *Consider an SVM-type objective as in (9). For any $\mathbf{w}_o$ and any $\delta > 0$, with probability at least $1-\delta$ over a sample of size $n$, we have that for all $\tilde{\mathbf{w}}$ s.t. $\hat{F}_{\lambda(n)}(\tilde{\mathbf{w}}) \leq \min \hat{F}_{\lambda(n)}(\mathbf{w}) + O(\frac{B^2}{\lambda n})$, where $\lambda(n)$ chosen as in (13), the following holds:*

$$\mathcal{L}(\tilde{\mathbf{w}}) \leq \mathcal{L}(\mathbf{w}_o) + O\left( \sqrt{\frac{B^2 \|\mathbf{w}_o\|^2 \log(1/\delta)}{n}} \right)$$

Corollary 4 is demonstrated empirically on the right plot of Figure 1.

The way we set $\lambda(n)$ in Corollary 4 depends on $\|\mathbf{w}_o\|$. However, using

$$\lambda(n) = \frac{B\sqrt{\log(1/\delta)}}{\sqrt{n}} \tag{14}$$

we obtain:

**Corollary 5.** *Consider an SVM-type objective as in (9) with $\lambda(n)$ set as in (14). For any $\delta > 0$, with probability at least $1 - \delta$ over a sample of size $n$, we have that for all $\tilde{\mathbf{w}}$ s.t. $\hat{F}_{\lambda(n)}(\tilde{\mathbf{w}}) \leq \min \hat{F}_{\lambda(n)}(\mathbf{w}) + O(\frac{B^2}{\lambda n})$, the following holds:*

$$\mathcal{L}(\tilde{\mathbf{w}}) \leq \inf_{\mathbf{w}_o} \left( \mathcal{L}(\mathbf{w}_o) + O\left( \sqrt{\frac{B^2(\|\mathbf{w}_o\|^4 + 1)\log(1/\delta)}{n}} \right) \right)$$

The price we pay here is that the bound of Corollary 5 is larger by a factor of $\|\mathbf{w}_o\|$ relative to the bound of Corollary 4. Nevertheless, this bound allows us to converge with a rate of $\sqrt{1/n}$ to the expected loss of *any* fixed predictor.

It is interesting to repeat the analysis of this Section using the more standard result:

$$F_\lambda(\mathbf{w}) - F_\lambda(\mathbf{w}^\star) \leq \hat{F}_\lambda(\mathbf{w}) - \hat{F}_\lambda(\mathbf{w}^\star) + O\left( \sqrt{\frac{B_{\mathbf{w}}^2 B^2}{n}} \right) \tag{15}$$

for $\|\mathbf{w}\| \leq B_{\mathbf{w}}$ where we ignore the dependence on $\delta$. Setting $B_{\mathbf{w}} = \sqrt{2/\lambda}$, as this is a bound on the norm of both the empirical and population optimums, and using (15) instead of Corollary 2 in our analysis yields the oracle inequality:

$$\mathcal{L}(\tilde{\mathbf{w}}) \leq \mathcal{L}(\mathbf{w}_o) + O\left( \left( \frac{B^2 \|\mathbf{w}_o\|^2 \log(1/\delta)}{n} \right)^{1/3} \right) \tag{16}$$

The oracle analysis studied here is very simple—our oracle assumption involves only a single predictor $\mathbf{w}_o$, and we make no assumptions about the kernel or the noise. We note that a more sophisticated analysis has been carried out by Steinwart *et al* [4], who showed that rates faster than $1/\sqrt{n}$ are possible under certain conditions on noise and complexity of kernel class. In Steinwart's *et al* analyses the estimation rates (i.e. rates for expected regularized risk) are given in terms of the approximation error quantity $\frac{\lambda}{2}\|\mathbf{w}^\star\|^2 + \mathcal{L}(\mathbf{w}^\star) - \mathcal{L}^*$ where $\mathcal{L}^*$ is the Bayes risk. In our result we consider the estimation rate for regularized objective independent of the approximation error.

## 4 Proof of Main Result

To prove Theorem 1 we use techniques of reweighing and peeling following Bartlett *et al* [5].

For each $\mathbf{w}$, we define $g_{\mathbf{w}}(\theta) = f(\mathbf{w}; \theta) - f(\mathbf{w}^{\star}; \theta)$, and so our goal is to bound the expectation of $g_{\mathbf{w}}$ in terms of its empirical average. We denote by $\mathcal{G} = \{g_{\mathbf{w}} | \mathbf{w} \in W\}$.

Since our desired bound is not exactly uniform, and we would like to pay different attention to functions depending on their expected sub-optimality, we will instead consider the following reweighted class. For any $r > 0$ define

$$\mathcal{G}_r = \left\{ g_{\mathbf{w}}^r = \frac{g_{\mathbf{w}}}{4^{k(\mathbf{w})}} : \mathbf{w} \in W, k(\mathbf{w}) = \min\{k' \in \mathbb{Z}_+ : \mathbb{E}\left[g_{\mathbf{w}}\right] \le r4^{k'}\} \right\} \tag{17}$$

where $\mathbb{Z}_+$ is the set of non-negative integers. In other words, $g_{\mathbf{w}}^r \in \mathcal{G}_r$ is just a scaled version of $g_{\mathbf{w}} \in \mathcal{G}$ and the scaling factor ensures that $\mathbb{E}\left[g_{\mathbf{w}}^r\right] \le r$.

We will begin by bounding the variation between expected and empirical average values of $g^r \in \mathcal{G}_r$. This is typically done in terms of the complexity of the class $\mathcal{G}_r$. However, we will instead use the complexity of a slightly different class of functions, which ignores the non-random (i.e. non-data-dependent) regularization terms $r(\mathbf{w})$. Define:

$$\mathcal{H}_r = \left\{ h_{\mathbf{w}}^r = \frac{h_{\mathbf{w}}}{4^{k(\mathbf{w})}} : \mathbf{w} \in \mathbf{W}, k(\mathbf{w}) = \min\{k' \in \mathbb{Z}_+ : \mathbb{E}\left[g_{\mathbf{w}}\right] \le r4^{k'}\} \right\} \tag{18}$$

where

$$h_{\mathbf{w}}(\theta) = g_{\mathbf{w}}(\theta) - (r(\mathbf{w}) - r(\mathbf{w}^{\star})) = \ell(\langle \mathbf{w}, \phi(\theta) \rangle; \theta) - \ell(\langle \mathbf{w}^*, \phi(\theta) \rangle; \theta). \tag{19}$$

That is, $h_{\mathbf{w}}^r(\theta)$ is the data dependent component of $g_{\mathbf{w}}^r$, dropping the (scaled) regularization terms. With this definition we have $\mathbb{E}\left[g_{\mathbf{w}}^r\right] - \hat{\mathbb{E}}\left[g_{\mathbf{w}}^r\right] = \mathbb{E}\left[h_{\mathbf{w}}^r\right] - \hat{\mathbb{E}}\left[h_{\mathbf{w}}^r\right]$ (the regularization terms on the left hand side cancel out), and so it is enough to bound the deviation of the empirical means in $\mathcal{H}_r$. This can be done in terms of the *Rademacher Complexity* of the class, $\mathcal{R}(\mathcal{H}_r)$ [6, Theorem 5]: For any $\delta > 0$, with probability at least $1 - \delta$,

$$\sup_{h^r \in \mathcal{H}_r} \mathbb{E}\left[h^r\right] - \hat{\mathbb{E}}\left[h^r\right] \le 2\mathcal{R}(\mathcal{H}_r) + \left( \sup_{h^r \in \mathcal{H}_r, \theta} |h^r(\theta)| \right) \sqrt{\frac{\log 1/\delta}{2n}}. \tag{20}$$

We will now proceed to bounding the two terms on the right hand side:

**Lemma 6.** $\displaystyle\sup_{h^r \in \mathcal{H}_r, \theta} |h^r(\theta)| \le LB\sqrt{2r/\lambda}$

*Proof.* From the definition of $h_{\mathbf{w}}^r$ given in (18)–(19), the Lipschitz continuity of $\ell(\cdot; \theta)$, and the bound $\|\phi(\theta)\|_* \le B$, we have for all $\mathbf{w}, \theta$:

$$|h_{\mathbf{w}}^r(\theta)| \le \frac{|h_{\mathbf{w}}(\theta)|}{4^{k(\mathbf{w})}} \le LB\|\mathbf{w} - \mathbf{w}^{\star}\|/4^{k(\mathbf{w})} \tag{21}$$

We now use the strong convexity of $F(\mathbf{w})$, and in particular eq. (7), as well as the definitions of $g_{\mathbf{w}}$ and $k(\mathbf{w})$, and finally note that $4^{k(\mathbf{w})} \ge 1$, to get:

$$\|\mathbf{w} - \mathbf{w}^{\star}\| \le \sqrt{\frac{2}{\lambda}(F(\mathbf{w}) - F(\mathbf{w}^{\star}))} = \sqrt{\frac{2}{\lambda}\mathbb{E}\left[g_{\mathbf{w}}\right]} \le \sqrt{\frac{2}{\lambda}4^{k(\mathbf{w})}r} \le \sqrt{\frac{2}{\lambda}16^{k(\mathbf{w})}r} \tag{22}$$

Substituting (22) in (21) yields the desired bound. □

**Lemma 7.** $\mathcal{R}(\mathcal{H}_r) \le 2LB\sqrt{\frac{2r}{\lambda n}}$

*Proof.* We will use the following generic bound on the Rademacher complexity of linear functionals [7, Theorem 1]: for any $t(\mathbf{w})$ which is $\lambda$-strongly convex (w.r.t a norm with dual norm $\|\cdot\|_*$),

$$\mathcal{R}(\{\phi \mapsto \langle \mathbf{w}, \phi \rangle \mid t(\mathbf{w}) \le a\}) \le (\sup \|\phi\|_*)\sqrt{\frac{2a}{\lambda n}}. \tag{23}$$

For each $a > 0$, define $\mathcal{H}(a) = \{h_{\mathbf{w}} : \mathbf{w} \in \mathbf{W}, \mathbb{E}\left[g_{\mathbf{w}}\right] \le a\}$. First note that $\mathbb{E}\left[g_{\mathbf{w}}\right] = F(\mathbf{w}) - F(\mathbf{w}^{\star})$ is $\lambda$-strongly convex. Using (23) and the Lipschitz composition property we therefore have $\mathcal{R}(\mathcal{H}(a)) \le LB\sqrt{\frac{2a}{\lambda n}}$. Now:

$$\mathcal{R}(\mathcal{H}_r) = \mathcal{R}\left(\cup_{j=0}^{\infty} 4^{-j}\mathcal{H}(r4^j)\right) \le \sum_{j=0}^{\infty} 4^{-j}\mathcal{R}(\mathcal{H}(4r^j)) \le LB\sqrt{\frac{2r}{\lambda n}}\sum_{j=0}^{\infty} 4^{-j/2} = 2LB\sqrt{\frac{2r}{\lambda n}} \quad \square$$

We now proceed to bounding $\mathbb{E}[g_\mathbf{w}] = F(\mathbf{w}) - F(\mathbf{w}^\star)$ and thus proving Theorem 1. For any $r > 0$, with probability at least $1 - \delta$ we have:

$$\mathbb{E}[g_\mathbf{w}] - \hat{\mathbb{E}}[g_\mathbf{w}] = 4^{k(\mathbf{w})}(\mathbb{E}[g_\mathbf{w}^r] - \hat{\mathbb{E}}[g_\mathbf{w}^r]) = 4^{k(\mathbf{w})}(\mathbb{E}[h_\mathbf{w}^r] - \hat{\mathbb{E}}[h_\mathbf{w}^r]) \leq 4^{k(\mathbf{w})}\sqrt{r}D \qquad (24)$$

where $D = LB\sqrt{\frac{1}{\lambda n}}(4\sqrt{2} + \sqrt{\log(1/\delta)}) \leq 2LB\sqrt{\frac{32 + \log(1/\delta)}{\lambda n}}$ is obtained by substituting Lemmas 6 and 7 into (20). We now consider two possible cases: $k(\mathbf{w}) = 0$ and $k(\mathbf{w}) > 0$.

The case $k(\mathbf{w}) = 0$ corresponds to functions with an expected value close to optimal: $\mathbb{E}[g_\mathbf{w}] \leq r$, i.e. $F(\mathbf{w}) \leq F(\mathbf{w}^\star) + r$. In this case (24) becomes:

$$\mathbb{E}[g_\mathbf{w}] \leq \hat{\mathbb{E}}[g_\mathbf{w}] + \sqrt{r}D \qquad (25)$$

We now turn to functions for which $k(\mathbf{w}) > 0$, i.e. with expected values further away from optimal. In this case, the definition of $k(\mathbf{w})$ ensures $4^{k(\mathbf{w})-1}r < \mathbb{E}[g_\mathbf{w}]$ and substituting this into (24) we have $\mathbb{E}[g_\mathbf{w}] - \hat{\mathbb{E}}[g_\mathbf{w}] \leq \frac{4}{r}\mathbb{E}[g_\mathbf{w}]\sqrt{r}D$. Rearranging terms yields:

$$\mathbb{E}[g_\mathbf{w}] \leq \frac{1}{1 - 4D/\sqrt{r}}\hat{\mathbb{E}}[g_\mathbf{w}] \qquad (26)$$

Combining the two cases (25) and (26) (and requiring $r \geq (4D)^2$ so that $\frac{1}{1-4D/\sqrt{r}} \geq 1$), we always have:

$$\mathbb{E}[g_\mathbf{w}] \leq \frac{1}{1-4D/\sqrt{r}}\left[\hat{\mathbb{E}}[g_\mathbf{w}]\right]_+ + \sqrt{r}D \qquad (27)$$

Setting $r = (1 + \frac{1}{a})^2(4D)^2$ yields the bound in Theorem 1.

## 5  Comparison with Previous "Fast Rate" Guarantees

Rates faster than $1/\sqrt{n}$ for estimation have been previously explored under various conditions, where strong convexity has played a significant role. Lee *et al* [8] showed faster rates for squared loss, exploiting the strong convexity of this loss function, but only under finite pseudo-dimensionality assumption, which do not hold in SVM-like settings. Bousquet [9] provided similar guarantees when the spectrum of the kernel matrix (covariance of the data) is exponentially decaying. Tsybakov [10] introduced a margin condition under which rates faster than $1/\sqrt{n}$ are shown possible. It is also possible to ensure rates of $1/n$ by relying on low noise conditions [9, 11], but here we make no such assumption.

Most methods for deriving fast rates first bound the variance of the functions in the class by some monotone function of their expectations. Then, using methods as in Bartlett *et al* [5], one can get bounds that have a localized complexity term and additional terms of order faster than $1/\sqrt{n}$. However, it is important to note that the localized complexity term typically dominates the rate and still needs to be controlled. For example, Bartlett *et al* [12] show that strict convexity *of the loss function* implies a variance bound, and provide a general result that can enable obtaining faster rates as long as the complexity term is low. For instance, for classes with finite VC dimension $V$, the resulting rate is $n^{-(V+2)/(2V+2)}$, which indeed is better than $1/\sqrt{n}$ but is not quite $1/n$. Thus we see that even for a strictly convex loss function, such as the squared loss, additional conditions are necessary in order to obtain "fast" rates.

In this work we show that strong convexity not only implies a variance bound but in fact can be used to bound the localized complexity. An important distinction is that we require strong convexity of the function $F(\mathbf{w})$ with respect to the norm $\|\mathbf{w}\|$. This is rather different than requiring the loss function $z \mapsto \ell(z, y)$ be strongly convex on the reals. In particular, the loss of a linear predictor, $\mathbf{w} \mapsto \ell(\langle \mathbf{w}, \mathbf{x} \rangle, y)$ can never be strongly convex in a multi-dimensional space, even if $\ell$ is strongly convex, since it is flat in directions orthogonal to $\mathbf{x}$.

As mentioned, $f(\mathbf{w}; \mathbf{x}, y) = \ell(\langle \mathbf{w}, \mathbf{x} \rangle, y)$ can never be strongly convex in a high-dimensional space. However, we actually only require the strong convexity of the expected loss $F(\mathbf{w})$. If the loss function $\ell(z, y)$ is $\lambda$-strongly convex in $z$, and the eigenvalues of the covariance of $\mathbf{x}$ are bounded away from zero, strong convexity of $F(\mathbf{w})$ can be ensured. In particular, $F(\mathbf{w})$ would be $c\lambda$-strongly-convex, where $c$ is the minimal eigenvalue of the $\mathrm{COV}[\mathbf{x}]$. This enables us to use Theorem

1 to obtain rates of $1/n$ on the expected loss itself. However, we cannot expect the eigenvalues to be bounded away from zero in very high dimensional spaces, limiting the applicability of the result of low-dimensional spaces were, as discussed above, other results also apply.

An interesting observation about our proof technique is that the only concentration inequality we invoked was McDiarmid's Inequality (in [6, Theorem 5] to obtain (20)—a bound on the deviations in terms of the Rademacher complexity). This was possible because we could make a localization argument for the $\ell_\infty$ norm of the functions in our function class in terms of their expectation.

## 6 Summary

We believe this is the first demonstration that, without any additional requirements, the SVM objective converges to its infinite data limit with a rate of $O(1/n)$. This improves the previous results that considered the SVM objective only under special additional conditions. The results extends also to other regularized objectives.

Although the quantity that is ultimately of interest to us is the expected loss, and not the *regularized* expected loss, it is still important to understand the statistical behavior of the regularized expected loss. This is the quantity that we actually optimize, track, and often provide bounds on (e.g. in approximate or stochastic optimization approaches). A better understanding of its behavior can allow us to both theoretically explore the behavior of regularized learning methods, to better understand empirical behavior observed in practice, and to appreciate guarantees of stochastic optimization approaches for such regularized objectives. As we saw in Section 3, deriving such fast rates is also essential for obtaining simple and general oracle inequalities, that also helps us guide our choice of regularization parameters.

## References

[1] E. Hazan, A. Kalai, S. Kale, and A. Agarwal. Logarithmic regret algorithms for online convex optimization. In *Proceedings of the Nineteenth Annual Conference on Computational Learning Theory*, 2006.

[2] S. Shalev-Shwartz. *Online Learning: Theory, Algorithms, and Applications*. PhD thesis, The Hebrew University, 2007.

[3] T. Zhang. Covering number bounds of certain regularized linear function classes. *J. Mach. Learn. Res.*, 2:527–550, 2002.

[4] I. Steinwart, D. Hush, and C. Scovel. A new concentration result for regularized risk minimizers. *High-dimensional Probability IV, in IMS Lecture Notes*, 51:260–275, 2006.

[5] P. L. Bartlett, O. Bousquet, and S. Mendelson. Localized rademacher complexities. In *COLT '02: Proceedings of the 15th Annual Conference on Computational Learning Theory*, pages 44–58, London, UK, 2002. Springer-Verlag.

[6] O. Bousquet, S. Boucheron, and G. Lugosi. Introduction to statistical learning theory. In O. Bousquet, U.v. Luxburg, and G. Rätsch, editors, *Advanced Lectures in Machine Learning*, pages 169–207. Springer, 2004.

[7] S. M. Kakade, K. Sridharan, and A. Tewari. On the complexity of linear prediction: Risk bounds, margin bounds, and regularization. In *NIPS*, 2008.

[8] W. S. Lee, P. L. Bartlett, and R. C. Williamson. The importance of convexity in learning with squared loss. In *Computational Learing Theory*, pages 140–146, 1996.

[9] O. Bousquet. *Concentration Inequalities and Empirical Processes Theory Applied to the Analysis of Learning Algorithms*. PhD thesis, Ecole Polytechnique, 2002.

[10] A. Tsybakov. Optimal aggregation of classifiers in statistical learning. *Annals of Statistics*, 32:135–166, 2004.

[11] I. Steinwart and C. Scovel. Fast rates for support vector machines using gaussian kernels. *ANNALS OF STATISTICS*, 35:575, 2007.

[12] P. L. Bartlett, M. I. Jordan, and J. D. McAuliffe. Convexity, classification, and risk bounds. *Journal of the American Statistical Association*, 101:138–156, March 2006.

